# Linear Time Inference in Hierarchical HMMs

**Kevin P. Murphy and Mark A. Paskin**
Computer Science Department
University of California
Berkeley, CA 94720-1776
{murphyk,paskin}@cs.berkeley.edu

## Abstract

The hierarchical hidden Markov model (HHMM) is a generalization of the hidden Markov model (HMM) that models sequences with structure at many length/time scales [FST98]. Unfortunately, the original inference algorithm is rather complicated, and takes $O(T^3)$ time, where $T$ is the length of the sequence, making it impractical for many domains. In this paper, we show how HHMMs are a special kind of dynamic Bayesian network (DBN), and thereby derive a much simpler inference algorithm, which only takes $O(T)$ time. Furthermore, by drawing the connection between HHMMs and DBNs, we enable the application of many standard approximation techniques to further speed up inference.

## 1   Introduction

The Hierarchical HMM [FST98] is an extension of the HMM that is designed to model domains with hierarchical structure, e.g., natural language, XML, DNA sequences [HIM$^+$00], handwriting [FST98], plan recognition [BVW00], visual action recogntion [IB00, ME01, Hoe01], and spatial navigation [TRM01, BVW01]. HHMMs are less expressive than stochastic context free grammars (SCFGs), since they only allows hierarchies of bounded depth, but they are more efficient and easier to learn. Unfortunately, the original inference algorithm[1] described in [FST98] is somewhat complicated, and takes $O(T^3 Q^D)$ time, where $T$ is the length of the sequence, $D$ is the depth of the hierarchy, and $Q$ is the (maximum) number of states at each level of the hierarchy. In this paper, we show how to represent an HHMM as a dynamic Bayesian network (DBN), and thereby derive a much simpler and faster inference algorithm, which takes at most $O(TQ^{2D})$ time; empirically, we find it takes only $O(TDQ^{\lceil 1.5D \rceil})$ time using the junction tree algorithm. Furthermore, by drawing the connection between HHMMs and DBNs, we enable the application of approximate inference techniques such as belief propagation, which can perform inference in $O(TD^2Q^2)$ time.

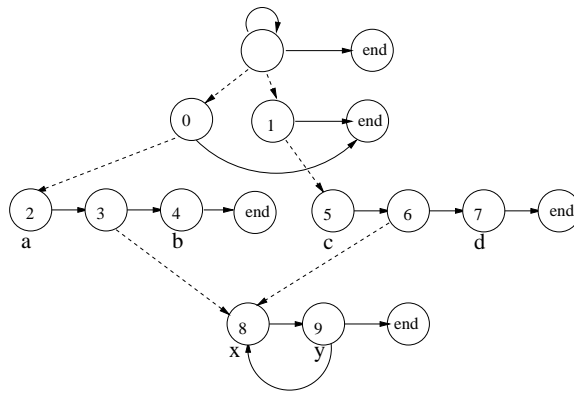

Figure 1: A 3-level hierarchical automaton representing the regular expression $a(xy)^+b|c(xy)^+d$. Solid lines represent horizontal transitions, dotted lines represent vertical transitions. Letters below a production state represent the symbol that is emitted. The unnumbered root node is considered level 0, and could be omitted if we fully interconnected states 0 and 1.

We will describe HHMMs in Section 2, and the original $O(T^3)$ inference algorithm in Section 3. The main contribution of the paper is in Section 4, where we show how to represent an HHMM as a DBN. In Section 5, we discuss how to do efficient inference in this DBN, and in Section 6, we discuss related work. In the full version of this paper, we discuss how to do parameter and structure learning using EM.

## 2 Hierarchical HMMs

HHMMs are like HMMs except the states of the stochastic automaton can emit single observations or strings of observations. (For simplicity of exposition, we shall assume all observations are discrete symbols, but HHMMs can easily be generalized to handle continuous observations, as we discuss in Section 4.1.) Those that emit single symbols are called "production states", and those that emit strings are termed "abstract states". The strings emitted by abstract states are themselves governed by sub-HHMMs, which can be called recursively. When the sub-HHMM is finished, control is returned to wherever it was called from; the calling context is memorized using a depth-limited stack.

We illustrate the generative process with Figure 1, which shows the state transition diagram of an example HHMM which models the regular expression $a(xy)^+b|c(xy)^+d$. We start in the root state, and make a "vertical transition" to one of its children, say state 0. From here, we make another vertical transition to state 2. Since state 2 is a production state, it emits "a" and then makes a "horizontal transition" to state 3. State 3 calls its sub-HMM, which emits x's and y's until it enters its end state; then control is returned to the calling state, in this case state 3. We then make a horizontal transition to state 4, emit "b", and enter the end state, thereby returning control to state 0. Finally, from state 0, we return control to the root, and optionally start again.

Any HHMM can be converted to an HMM by creating a state for every possible legal stack configuration $Q_t^{1:D}$. If the HHMM transition diagram is a tree, there will be one HMM state for every HHMM production state. If the HHMM transition diagram has shared substructure (such as the sub-expression $(xy)^+$), this structure must be duplicated in the HMM, generally resulting in a larger model. It is the ability to reuse sub-models in different con-

texts that makes HHMMs more powerful than standard HMMs. In particular, the parameters of such shared sub-models only need to be learned once. (Given segmented data, we can train the sub-HMMs separately, and then "glue them together", but it is also possible to train the HHMM on unsegmented data; see the full version of this paper for details.)

## 3   Cubic-time inference

The inference algorithm for HHMMs presented in [FST98] runs in $O(T^3)$ time and is based on the Inside-Outside algorithm [LY90], an exact inference algorithm for stochastic context-free grammars (SCFGs) which we now describe.

In an SCFG, sequences of observations are generated by a set of stochastic production rules. Each production rule stochastically rewrites a non-terminal symbol $N^i$ into either a symbol of the alphabet ($N^i \overset{p}{\to} a$) or a pair of nonterminal symbols ($N^i \overset{p}{\to} N^j N^k$). Observation strings are generated by starting with the distinguished "start" nonterminal $N^0$, and continually re-writing all non-terminals using stochastic production rules until, finally, only symbols of the alphabet remain.

The Inside-Outside algorithm computes $P(N^i \to N^j N^k | O_{t:t+k})$, where $O_{t:t+k} = O_t, O_{t+1}, \ldots, O_{t+k}$ is a subsequence. This can then be used to compute the expected sufficient statistics needed by the EM algorithm to learn the parameters of the model. If there are $N$ non-terminals in the grammar and the training sequence is of length $T$, then the Inside-Outside algorithm requires $O(N^3 T^3)$ time. To see why, note that we must compute $P(N^i \to N^j N^k | O_{t:t+\tau})$ for all end points $t$ and $t + \tau$, and for all midpoints $\tau'$ such that $N^j$ generates $O_{t:t+\tau'}$ and $N^k$ generates $O_{t+\tau'+1:t+\tau}$ — the three degrees for freedom $t, \tau$ and $\tau'$ gives rise to the $T^3$ term. The $N^3$ term arises because we must consider all $N^i$, $N^j$ and $N^k$.

The inference algorithm for HHMMs presented in [FST98] is based upon a straightforward adaptation of the Inside-Outside algorithm. The algorithm computes $P(\text{in state } q_i^d \text{ at time } t | O_{t:t+\tau})$ by assuming that sub-state $q_j^{d+1}$ generates $O_{t:t+\tau'}$, that a transition to state $k$ occurs, and that $q_k^{d+1}$ generates $O_{t+\tau'+1:t+\tau}$. Hence the algorithm takes $O(NT^3)$ time, where $N$ is the total number of states.

We can always "flatten" an HHMM into a regular HMM and hence do inference in $O(N^2 T)$. Unfortunately, this flat model cannot represent the hierarchical structure, yet alone learn it. In the next section, we show how to represent the HHMM as a DBN, and thereby get the best of both worlds: low time complexity without losing hierarchical structure.

## 4   Representing the HHMM as a DBN

We can represent the HHMM as a dynamic Bayesian network (DBN) as shown in Figure 2. (We assume for simplicity that all production states are at the bottom of the hierarchy; this restriction is lifted in the full version of this paper.) The state of the HMM at level $d$ and time $t$ is represented by $Q_t^d$. The state of the whole HHMM is encoded by the vector $\vec{Q}_t = (Q_t^1, \ldots, Q_t^P)$; intuitively, this encodes the contents of the stack, that specifies the complete "path" to take from the root to the leaf state.

$F_t^d$ is an indicator variable that is "on" if the HMM at level $d$ and time $t$ has just "finished" (i.e., is about to enter an end state), otherwise it is off. Note that if $F_t^d = 1$, then $F_t^{d'} = 1$

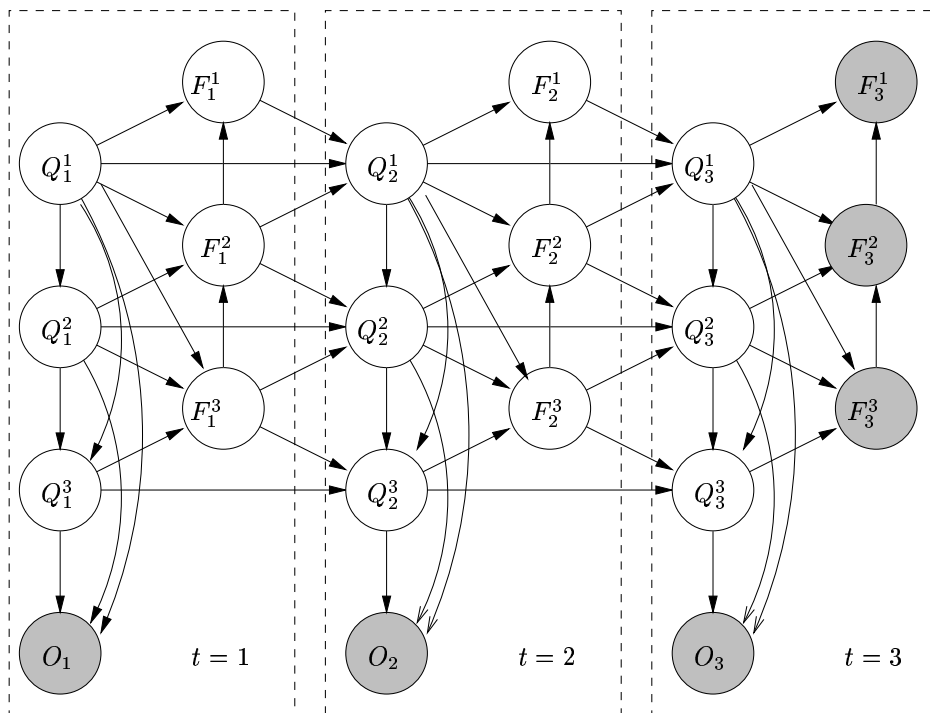

Figure 2: An HHMM represented as a DBN. $Q_t^d$ is the state at time $t$, level $d$; $F_t^d = 1$ if the HMM at level $d$ has finished (entered its exit state), otherwise $F_t^d = 0$. Shaded nodes are observed; the remaining nodes are hidden. We may optionally clamp $F_T^d = 1$, where $T$ is the length of the observation sequence, to ensure all models have finished by the end of the sequence. (A similar trick was used in [Zwe97].)

for all $d' > d$; hence the number of $F$ nodes that are "off" represents the effective height of the "context stack", i.e., which level of the hierarchy we are currently on.

The downward going arcs between the $Q$ variables represent the fact that a state "calls" a sub-state. The upward going arcs between the $F$ variables enforce the fact that a higher-level HMM can only change state when the lower-level one is finished. This ensures proper nesting of the parse trees, and is the key difference between an HHMM and a hidden Markov decision tree [JGS96].

We will define the conditional probability distributions (CPDs) of each of the node types below, which will complete the definition of the model. We consider the bottom, middle and top layers of the hierarchy separately (since they have different local topology), as well as the first, middle and last time slices.

## 4.1 Definition of the CPDs

Consider the bottom level of the hierarchy. $Q^D$ follows a Markov chain with parameters determined by its position in the automaton, which is encoded by the vector of higher-up state variables $Q_t^{1:D-1} = Q_t^1, \ldots, Q_t^{D-1}$, which we will represent by the integer $k$.[2] When

$Q^D$ enters its end state, it will "turn on" $F^D$, to mean it is finished; this will be a signal that higher-level HMMs can now change state. In addition, it will be a signal that the next value of $Q^D$ should be drawn from its prior distribution (representing a vertical transition), instead of its transition matrix (representing a horizontal transition). Formally, we can write this as follows:

$$P(Q_t^D = j | Q_{t-1}^D = i, F_{t-1}^D = f, Q_t^{1:D-1} = k) = \begin{cases} \tilde{A}_k^D(i, j) & \text{if } f = 0 \\ \pi_k^D(j) & \text{if } f = 1 \end{cases}$$

where we have assumed $i, j \neq$ end. $A_k^D$ is the transition matrix for level $D$ given that the parent variables are in state $k$, and $\tilde{A}_k^D$ is just a rescaled version of $A_k^D$.[3] Similarly, $\pi_k^D$ is the initial distribution for level $D$ given that the parent variables are in state $k$. The equation for $F_D$ is simply

$$P(F_t^D = 1 | Q_t^{1:D-1} = k, Q_t^D = i) = A_k^D(i, \text{end}).$$

Now consider the intermediate levels. As before, $Q^d$ follows a Markov chain with parameters determined by $Q^{1:d-1}$, and $F^d$ specifies whether we should use the transition matrix or the prior. The difference is that we now also get a signal from below, $F^{d+1}$, specifying whether the sub-model has finished or not; if it has, we are free to change state, otherwise we must remain in the same state. Formally, we can write this as follows:

$$P(Q_t^d = j | Q_{t-1}^d = i, F_{t-1}^{d+1} = b, F_{t-1}^d = f, Q_t^{1:d-1} = k) = \begin{cases} \delta(i, j) & \text{if } b = 0 \\ \tilde{A}_k^d(i, j) & \text{if } b = 1 \text{ and } f = 0 \\ \pi_k^d(j) & \text{if } b = 1 \text{ and } f = 1 \end{cases}$$

$F^d$ should "turn on" only if $Q^d$ is "allowed" to enter a final state, the probability of which depends on the current context $Q^{1:d-1}$. Formally, we can write this as follows:

$$P(F_t^d = 1 | Q_t^d = i, Q_t^{1:d-1} = k, F_t^{d+1} = b) = \begin{cases} 0 & \text{if } b = 0 \\ A_k^d(i, \text{end}) & \text{if } b = 1 \end{cases}$$

The top level differs from the intermediate levels in that the $Q$ node has no $Q$ parent to specify which distribution to use. The equations are the same as above, except we eliminate the conditioning on $Q_t^{1:d-1} = k$. (Equivalently, we can imagine a dummy top layer HMM, which is always in state 1: $Q_t^0 = 1$. This is often how HHMMs are represented, so that this top-level state is the root of the overall parse tree, as in Figure 1.)

The CPDs for the nodes in the first slice are as follows: $P(Q_1^1 = j) = \pi^1(j)$ for the top level and $P(Q_1^d = j | Q_1^{1:d-1} = k) = \pi_k^d(j)$, for $d = 2, \ldots, D$.

If the observations are discrete symbols, we may represent $P(O_t | \vec{Q}_t)$ as a multinomial (i.e., using a table), or by using any of the more parsimonious representations discussed in Section 4.2. If the observations are real-valued vectors, we can use a Gaussian for each value of $\vec{Q}_t$, or a mixture of a smaller number of Gaussians, as in [GJ97].

[3]Unlike the automaton representation, the DBN never actually enters an end state (i.e., $Q_t^d$ can never taken on the value 'end'), because if it did, it would not be able to emit the symbol $O_t$. Instead, $Q_t^d$ causes $F_t^d$ to turn on, and then enters a new (non-terminal) state at time $t + 1$. This means that the DBN and HHMM transition matrices are not identical, but satisfy the following relation: $\tilde{A}_k^d(i, j) \left(1 - \tau_k^d(i)\right) = A_k^d(i, j)$, where $A$ represents the automaton transition matrix, $\tilde{A}$ represents the DBN transition matrix, and $\tau_k^d(i) \stackrel{\text{def}}{=} A_k^d(i, \text{end})$ is the probability of terminating from state $i$. The equations holds because the probability of each horizontal transition in the DBN gets multiplied by the probability that $F_t^d = 0$, which is $1 - \tau_k^d(i)$; this product should match the original probability. It is easy to see that the new matrix is also stochastic, as required.

## 4.2 Parsimonious representations of the CPDs

The number of parameters needed to represent $P(Q_t^d|Q_{t-1}^d, Q_t^{1:d-1} = k)$ as a multinomial is $O(Q^{d+1})$. If the state-transition diagram of the hierarchical automaton is sparse, many of the entries in this table will be 0. However, when we are learning a model, we do not know the structure of the state-transition diagram, and must therefore adopt a representation with fewer parameters. There are at least three possibilities: decision trees [BFGK96], softmax nodes, or representing $P(Q_t^d|Q_{t-1}^d, Q_t^{1:d-1} = k)$ as a mixture of smaller transition matrices at different depths c.f. [SJ99]. See the full version of this paper for details.

## 5 Linear-time inference

We define inference to be computing $P(S_i|O_{1:T})$ for all sets of nodes $S_i = \{i\} \cup \text{parents}(i)$ in the DBN. These "family" marginals are needed by EM. The simplest way to do this is to merge all the hidden nodes in each slice into a single "mega node", $M_t$, with $M = 2^D Q^D$ possible values. (The $2^D$ term arises from the binary $F$ nodes.) We can then apply the forwards-backwards algorithm for HMMs, which takes $O(M^2 T)$ time.

Unfortunately, converting the DBN to an HMM in this way will not be tractable for reasonably large $Q$ or $D$. (Even storing the $M \times M$ transition matrix is likely to consume too much space.) Fortunately, we can do better by exploiting the structure of the model. In [Mur01], we present a way of applying the junction tree (jtree) algorithm to variable-length DBNs; we give a brief sketch here. The algorithm works by performing a forwards-backwards sweep through a chain of jtrees. Each jtree is formed from a "$1\frac{1}{2}$-slice DBN"; this is a DBN that contains all the nodes in slice 1 but only the interface nodes from slice 2. The interface nodes are those nodes in slice 2 that have an incoming temporal arc, plus parents of nodes that have incoming temporal arcs. In the case of an HHMM, the interface is all the $Q$ nodes.

The cost of doing inference in each jtree depends on the sizes of the cliques. Minimizing the maximal clique size is NP-hard, so we used a standard one-step look-ahead (greedy) algorithm [Kja90]. The resulting cliques are hard to interpret, but we can still analyze the complexity. Let $N_Q(c, D)$ be the number of $Q$ nodes in clique $c$, let $N_F(c, D)$ be the number of $F$ nodes, and let $N_c(D)$ be the number of cliques. Then the cost of inference in a jtree is proportional to

$$\sum_{c=1}^{N_c(D)} Q^{N_Q(c,D)} \times 2^{N_F(c,D)} < N_c(D) \times Q^{\max_c N_Q(c,D)} \times 2^{\max_c N_F(c,D)}$$

Empirically we find that, for a wide range of $D$, $N_c(D) < D + 2$, $\max_c N_F(c, D) \leq \lceil 0.5D \rceil$ and $\max_c N_Q(c, D) \leq 1 + \lceil 1.5D \rceil$. Hence a crude upper bound on the cost of inference in each jtree is $O((D+2)Q^{\lceil 1.5D \rceil} 2^{\lceil 0.5D \rceil})$, yielding an overall time and space complexity of $O(TDQ^{\lceil 1.5D \rceil})$. We remind readers that the original algorithm has $O(T^3 Q^D)$ time complexity, since there can be up to $N = Q^D$ states in the HHMM. The advantage of the new algorithm in practice is clearly illustrated in Figure 3.

We can reduce the time (and space) complexity from $O(TDQ^{\lceil 1.5D \rceil})$ to $O(TDQ^D)$ by using *approximate* DBN inference techniques such as the "factored frontier (FF) algorithm" [MW01], which is equivalent to applying "loopy belief propagation" to the DBN using a left-right scheduling of the messages. (It is still exponential in $D$ because of the high fan-in of the nodes.) We can get a further speedup by using a mixture representation of the CPDs

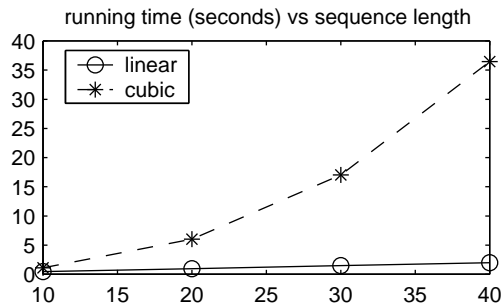

Figure 3: Running time vs. sequence length. Both algorithms were implemented in Matlab. The HHMM has $D = 2, Q = 3$.

(see Section 4.2). In this case, we can exploit the form of the CPD to compute the required messages efficiently [Mur99], bringing the overall complexity down to $O(TD^2Q^2)$.

We remark that all of the above algorithms can also be used for online filtering. In addition, by replacing the sum operator with max, we can do Viterbi segmentation in the usual way.

## 6 Related work

Hidden Markov decision trees (HMDT) [JGS96] are DBNs with a structure similar to Figure 2, but they lack the $F$ nodes and the upward going arcs; hence they are not able to represent the call-return semantics of the HHMM. Embedded HMMs [NI00] are a special case of HHMMs in which the ending "time" of the sub-HMMs is known in advance (e.g., the sub-HMM models exactly one row of pixels). ([Hoe01] calls these models "hierarchical mixture of Markov chains".) A variable-duration HMM [Rab89] is a special case of a 2-level HHMM, where the bottom level counts how long we have been in a certain state; when the counter expires, the $F$ node turns on, and the parent can change state.

[BVW00] describes the "Abstract HMM" (AHMM), which is very closely related to HHMMs. These authors are interested in inferring what abstract policy an agent is following by observing its effects in the world. An AHMM is equivalent to an HHMM if we consider $Q_t^d$ to represent the (abstract) policy being followed at level $d$ and time $t$; $Q_t^D$ represents the concrete action, which causes the observation. We also need to add a hidden global state variable $S_t$, which is a parent of the $O_t$ node, all the $F_t$ nodes and all the $Q_{t+1}$ nodes. ($S_t$ is hidden to us as observers, but not to the agent performing the actions.) [BVW00] consider abstract policies of the "options" kind [SPS99], which is equivalent to assuming that there are no horizontal transitions. (HAMs [PR97] generalize this by allowing horizontal transitions (i.e., internal state) within a controller.) In addition, they assume that $Q_t^d$ only depends on its immediate parent, $Q_t^{d-1}$, but not its whole context, $Q_t^{1:d-1}$, so the $Q$ nodes become connected by a chain. This enables them to use Rao-Blackwellized particle filtering for approximate online inference: conditioned on the $F$ nodes, the distribution over the $Q$ nodes can be represented as a product of marginals, so they can be efficiently marginalized out.

## Acknowledgements

I would like to thank Dr Christopher Schlick for giving me his Matlab implementation of the $O(T^3)$ algorithm, which was used to create part of Figure 3.

## Footnotes

[1] By inference, we mean offline smoothing, i.e., conditioning on a fixed-length observation sequence. This is needed as a subroutine for EM. Once the model has been learned, it will typically be used for online inference (filtering).

[2]If the topology is sparse, this distribution will be 0 for many values of $k$. This will be discussed in Section 4.2.

## References

[BFGK96] C. Boutilier, N. Friedman, M. Goldszmidt, and D. Koller. Context-Specific Independence in Bayesian Networks. In *UAI*, 1996.

[BVW00] H. Bui, S. Venkatesh, and G. West. On the recognition of abstract Markov policies. In *AAAI*, 2000.

[BVW01] H. Bui, S. Venkatesh, and G. West. Tracking and surveillance in wide-area spatial environments using the Abstract Hidden Markov Model. *Intl. J. of Pattern Rec. and AI*, 2001.

[FST98] Shai Fine, Yoram Singer, and Naftali Tishby. The hierarchical Hidden Markov Model: Analysis and applications. *Machine Learning*, 32:41, 1998.

[GJ97] Z. Ghahramani and M. Jordan. Factorial hidden Markov models. *Machine Learning*, 29:245–273, 1997.

[HIM+00] M. Hu, C. Ingram, M.Sirski, C. Pal, S. Swamy, and C. Patten. A Hierarchical HMM Implementation for Vertebrate Gene Splice Site Prediction. Technical report, Dept. Computer Science, Univ. Waterloo, 2000.

[Hoe01] J. Hoey. Hierarchical unsupervised learning of facial expression categories. In *ICCV Workshop on Detection and Recognition of Events in Video*, 2001.

[IB00] Y. Ivanov and A. Bobick. Recognition of visual activities and interactions by stochastic parsing. *IEEE Trans. on Pattern Analysis and Machine Intelligence*, 22(8):852–872, 2000.

[JGS96] M. I. Jordan, Z. Ghahramani, and L. K. Saul. Hidden Markov decision trees. In *NIPS*, 1996.

[Kja90] U. Kjaerulff. Triangulation of graphs – algorithms giving small total state space. Technical Report R-90-09, Dept. of Math. and Comp. Sci., Aalborg Univ., Denmark, 1990.

[LY90] K. Lari and S. J. Young. The estimation of stochastic context-free grammars using the Inside-Outside algorithm. *Computer Speech and Language*, 4:35–56, 1990.

[ME01] D. Moore and I. Essa. Recognizing multitasked activities using stochastic context-free grammar. In *CVPR Workshop on Models vs Exemplars in Computer Vision*, 2001.

[Mur99] K. Murphy. Pearl's algorithm and multiplexer nodes. Technical report, U.C. Berkeley, Dept. Comp. Sci., 1999.

[Mur01] K. Murphy. Applying the junction tree algorithm to variable-length DBNs. Technical report, Comp. Sci. Div., UC Berkeley, 2001.

[MW01] K. Murphy and Y. Weiss. The Factored Frontier Algorithm for Approximate Inference in DBNs. In *UAI*, 2001.

[NI00] A. Nefian and M. Hayes III. Maximum likelihood training of the embedded HMM for face detection and recognition. In *IEEE Intl. Conf. on Image Processing*, 2000.

[PR97] R. Parr and S. Russell. Reinforcement learning with hierarchies of machines. In *NIPS*, 1997.

[Rab89] L. R. Rabiner. A tutorial on Hidden Markov Models and selected applications in speech recognition. *Proc. of the IEEE*, 77(2):257–286, 1989.

[SJ99] L. Saul and M. Jordan. Mixed memory markov models: Decomposing complex stochastic processes as mixture of simpler ones. *Machine Learning*, 37(1):75–87, 1999.

[SPS99] R.S. Sutton, D. Precup, and S. Singh. Between MDPs and semi-MDPs: A framework for temporal abstraction in reinforcement learning. *Artificial Intelligence*, 112:181–211, 1999.

[TRM01] G. Theocharous, K. Rohanimanesh, and S. Mahadevan. Learning Hierarchical Partially Observed Markov Decision Process Models for Robot Navigation. In *ICRA*, 2001.

[Zwe97] G. Zweig. *Speech Recognition with Dynamic Bayesian Networks*. PhD thesis, U.C. Berkeley, Dept. Comp. Sci., 1997.